# Feature Construction for Inverse Reinforcement Learning

**Sergey Levine**
Stanford University
svlevine@cs.stanford.edu

**Zoran Popović**
University of Washington
zoran@cs.washington.edu

**Vladlen Koltun**
Stanford University
vladlen@cs.stanford.edu

## Abstract

The goal of inverse reinforcement learning is to find a reward function for a Markov decision process, given example traces from its optimal policy. Current IRL techniques generally rely on user-supplied features that form a concise basis for the reward. We present an algorithm that instead constructs reward features from a large collection of component features, by building logical conjunctions of those component features that are relevant to the example policy. Given example traces, the algorithm returns a reward function as well as the constructed features. The reward function can be used to recover a full, deterministic, stationary policy, and the features can be used to transplant the reward function into any novel environment on which the component features are well defined.

## 1 Introduction

Inverse reinforcement learning aims to find a reward function for a Markov decision process, given only example traces from its optimal policy. IRL solves the general problem of apprenticeship learning, in which the goal is to learn the policy from which the examples were taken. The MDP formalism provides a compact method for specifying a task in terms of a reward function, and IRL further simplifies task specification by requiring only a demonstration of the task being performed. However, current IRL methods generally require not just expert demonstrations, but also a set of *features* or *basis functions* that concisely capture the structure of the reward function [1, 7, 9, 10].

Incorporating feature construction into IRL has been recognized as an important problem for some time [1]. It is often easier to enumerate all potentially relevant component features ("components") than to manually specify a set of features that is both complete and fully relevant. For example, when emulating a human driver, it is easier to list all known aspects of the environment than to construct a complete and fully relevant reward basis. The difficulty of performing IRL given only such components is that many of them may have important logical relationships that make it impossible to represent the reward function as their linear combination, while enumerating all possible relationships is intractable. In our example, some of the components, like the color of the road, may be irrelevant. Others, like the car's speed and the presence of police, might have an important logical relationship for a driver who prefers to speed.

We present an IRL algorithm that constructs reward features out of a large collection of component features, many of which may be irrelevant for the expert's policy. The Feature construction for Inverse Reinforcement Learning (FIRL) algorithm constructs features as logical conjunctions of the components that are most relevant for the observed examples, thus capturing their logical relationships. At the same time, it finds a reward function for which the optimal policy matches

the examples. The reward function can be used to recover a deterministic, stationary policy for the expert, and the features can be used to transplant the reward to any novel environment on which the component features are well defined. In this way, the features act as a portable explanation for the expert's policy, enabling the expert's behavior to be predicted in unfamiliar surroundings.

## 2 Algorithm Overview

We define a Markov decision process as $\mathcal{M} = \{\mathcal{S}, \mathcal{A}, \theta, \gamma, R\}$, where $\mathcal{S}$ is a state space, $\mathcal{A}$ is a set of actions, $\theta_{sas'}$ is the probability of a transition from $s \in \mathcal{S}$ to $s' \in \mathcal{S}$ under action $a \in \mathcal{A}$, $\gamma \in [0, 1)$ is a discount factor, and $R(s, a)$ is a reward function. The optimal policy $\pi^*$ is the policy that maximizes the expected discounted sum of rewards $E\left[\sum_{t=0}^{\infty} \gamma^t R(s_t, a_t) | \pi^*, \theta\right]$. FIRL takes as input $\mathcal{M} \setminus R$, as well as a set of traces from $\pi^*$, denoted by $\mathcal{D} = \{(s_{1,1}, a_{1,1}), ..., (s_{n,T}, a_{n,T})\}$, where $s_{i,t}$ is the $t^{\text{th}}$ state in the $i^{\text{th}}$ trace. FIRL also accepts a set of component features of the form $\delta : \mathcal{S} \to \mathbb{Z}$, which are used to construct a set of relevant features for representing $R$.

The algorithm iteratively constructs both the features and the reward function. Each iteration consists of an optimization step and a fitting step. The algorithm begins with an empty feature set $\Phi^{(0)}$. The optimization step of the $i^{\text{th}}$ iteration computes a reward function $R^{(i)}$ using the current set of features $\Phi^{(i-1)}$, and the following fitting step determines a new set of features $\Phi^{(i)}$.

The objective of the optimization step is to find a reward function $R^{(i)}$ that best fits the last feature hypothesis $\Phi^{(i-1)}$ while remaining consistent with the examples $\mathcal{D}$. This appears similar to the objective of standard IRL methods. However, prior IRL algorithms generally minimize some measure of deviation from the examples, subject to the constraints of the provided features [1, 7, 8, 9, 10]. In contrast, the FIRL optimization step aims to discover regions where the current features are insufficient, and must be able to step outside of the constraints of the these features. To this end, the reward function $R^{(i)}$ is found by solving a quadratic program, with constraints that keep $R^{(i)}$ consistent with $\mathcal{D}$, and an objective that penalizes the deviation of $R^{(i)}$ from its projection onto the linear basis formed by the features $\Phi^{(i-1)}$.

The fitting step analyzes the reward function $R^{(i)}$ to generate a new feature hypothesis $\Phi^{(i)}$ that better captures the variation in the reward function. Intuitively, the regions where $R^{(i)}$ is poorly represented by $\Phi^{(i-1)}$ correspond to features that must be refined further, while regions where different features take on similar rewards are indicative of redundant features that should be merged. The hypothesis is constructed by building a regression tree on $\mathcal{S}$ for $R^{(i)}$, with the components acting as tests at each node. Each leaf $\ell$ contains some subset of $\mathcal{S}$, denoted $\phi_\ell$. The new features are the set of indicator functions for membership in $\phi_\ell$. A simple explanation of the reward function is often more likely to be the correct one [7], so we prefer the smallest tree that produces a sufficiently rich feature set to represent a reward function consistent with the examples. To obtain such a tree, we stop subdividing a node $\ell$ when setting the reward for all states in $\phi_\ell$ to their average induces an optimal policy consistent with the examples.

The constructed features are iteratively improved through the interaction between the optimization and fitting steps. Since the optimization is constrained to be consistent with $\mathcal{D}$, if the current set of features is insufficient to represent a consistent reward function, $R^{(i)}$ will not be well-represented by the features $\Phi^{(i-1)}$. This intra-feature reward variance is detected in the fitting step, and the features that were insufficiently refined are subdivided further, while redundant features that have little variance between them are merged.

## 3 Optimization Step

During the $i^{\text{th}}$ optimization step, we compute a reward function $R^{(i)}$ using the examples $\mathcal{D}$ and the current feature set $\Phi^{(i-1)}$. This reward function is chosen so that the optimal policy under the reward is consistent with the examples $\mathcal{D}$ and so that it minimizes the sum of squared errors between $R^{(i)}$ and its projection onto the linear basis of features $\Phi^{(i-1)}$. Formally, let $T_{R \to \Phi}$ be a $|\Phi^{(i-1)}|$ by $|\mathcal{S}|$ matrix for which $T_{R \to \Phi}(\phi, s) = |\phi|^{-1}$ if $s \in \phi$, and 0 otherwise, and let $T_{\Phi \to R}$ be a $|\mathcal{S}|$ by $|\Phi^{(i-1)}|$ matrix for which $T_{\Phi \to R}(s, \phi) = 1$ if $s \in \phi$, and 0 otherwise. Thus, $T_{\Phi \to R} T_{R \to \Phi} R$ is a vector where

the reward in each state is the average over all rewards in the feature that state belongs to. Letting $\pi^R$ denote the optimal policy under $R$, the reward optimization problem can be expressed as:

$$\min_R \|R - T_{\Phi \to R} T_{R \to \Phi} R\|^2$$

$$\text{s.t.} \quad \pi^R(s) = a \qquad \forall\, (s,a) \in \mathcal{D} \tag{1}$$

Unfortunately, the constraint (1) is not convex, making it difficult to solve the optimization efficiently. We can equivalently express it in terms of the value function corresponding to $R$ as

$$V(s) = R(s,a) + \gamma \sum_{s'} \theta_{sas'} V(s') \qquad\qquad \forall\, (s,a) \in \mathcal{D}$$

$$V(s) = \max_a R(s,a) + \gamma \sum_{s'} \theta_{sas'} V(s') \qquad\qquad \forall\, s \in \mathcal{S} \tag{2}$$

These constraints are also not convex, but we can construct a convex relaxation by using a pseudo-value function that bounds the value function from above, replacing (2) with the linear constraint

$$V(s) \geq R(s,a) + \gamma \sum_{s'} \theta_{sas'} V(s') \qquad\qquad \forall\, s \notin \mathcal{D}$$

In the special case that the MDP transition probabilities $\theta$ are deterministic, these constraints are equivalent to the original constraint (1). We prove this by considering the true value function $V^*$ obtained by value iteration, initialized with the pseudo-value function $V$. Let $V'$ be the result obtained by performing one step of value iteration. Note that $V'(s) \leq V(s)$ for all $s \in \mathcal{S}$: since $V(s) \geq R(s,a) + \gamma \sum_{s'} \theta_{sas'} V(s')$, we must have $V(s) \geq \max_a [R(s,a) + \gamma \sum_{s'} \theta_{sas'} V(s')] = V'(s)$. Since the MDP is deterministic and the example set $\mathcal{D}$ consists of traces from the optimal policy, we have a unique next state for each state-action pair. Let $(s_{i,t}, a_{i,t}) \in \mathcal{D}$ be the $t^{\text{th}}$ state-action pair from the $i^{\text{th}}$ expert trace. Since the constraints ensure that $V(s_{i,t}) = \max_a [R(s_{i,t},a) + \gamma V(s_{i,t+1})]$, we have $V'(s_{i,t}) = V(s_{i,t})$ for all $i,t$, and since $V'(s)$ for $s \notin \mathcal{D}$ can only decrease, we know that the optimal actions in all $s_{i,t}$ must remain the same. Therefore, for each example state $s_{i,t}$, $a_{i,t}$ remains the optimal action under the true value function $V^*$, and the convex relaxation is equivalent to the original constraint (1).

In the case that $\theta$ is not deterministic, not all successors of an example state $s_{i,t}$ are always observed, and their values under the pseudo-value function may not be sufficiently constrained. However, empirical tests presented in Figure 2(b) suggest that the constraint (1) is rarely violated under the convex relaxation, even in highly non-deterministic MDPs.

In practice, we prefer a reward function under which the examples are not just part of an optimal policy, but are part of the *unique* optimal policy [7]. To prevent rewards under which example actions "tie" for the optimal choice, we require that $a_{i,t}$ be better than all other actions in state $s_{i,t}$ by some margin $\varepsilon$, which we accomplish by adding $\varepsilon$ to all inequality constraints for state $s_{i,t}$. The precise value of $\varepsilon$ is not important, since changing it only scales the reward function by a constant.

All of the constraints in the final optimization are sparse, but the matrix $T_{\Phi \to R} T_{R \to \Phi}$ in the original objective can be arbitrarily dense (if, for instance, there is only one feature which contains all states). Since both $T_{\Phi \to R}$ and $T_{R \to \Phi}$ are sparse, and in fact only contain $|\mathcal{S}||\mathcal{A}|$ non-zero entries, we can make the optimization fully sparse by introducing a new set of variables $R_\Phi$ defined as $R_\Phi = T_{R \to \Phi} R$, yielding the sparse objective $\|R - T_{\Phi \to R} R_\Phi\|^2$.

Recall that the fitting step must determine not only which features must be refined further, but also which features can be merged. We therefore add a second term to the objective to discourage nearby features from taking on different values when it is unnecessary. To that end, we construct a sparse matrix $N$, where each row $k$ of $N$ corresponds to a pair of features $\phi_{k_1}$ and $\phi_{k_2}$ (for a total of $K$ rows). We define $N$ as $N_{k,\phi_{k_1}} = -N_{k,\phi_{k_2}} = \Delta(\phi_{k_1}, \phi_{k_2})$, so that $[NR_\Phi]_k = (R_{\Phi \phi_{k_1}} - R_{\Phi \phi_{k_2}})\Delta(\phi_{k_1}, \phi_{k_2})$. The loss factor $\Delta(\phi_{k_1}, \phi_{k_2})$ indicates how much we believe a priori that the features $\phi_{k_1}$ and $\phi_{k_2}$ should be merged, and is discussed further in Section 4. Since the purpose of the added term is to allow superfluous features to be merged because they take on similar values, we prefer for a feature to be very similar to one of its neighbors, rather than to have minimal distance to all of them. We therefore use a linear rather than quadratic penalty. Since we would like to make nearby features similar so long as it does not adversely impact the primary objective, we give this adjacency penalty a low weight. In our implementation, this weight was set to

$w_N = 10^{-5}$. Normalizing the two objectives by the number of entries, we get the following sparse quadratic program:

$$\min_{R, R_\Phi, V} \quad \frac{1}{|\mathcal{S}||\mathcal{A}|} \|R - T_{\Phi \to R} R_\Phi\|_2^2 + \frac{w_N}{K} \|N R_\Phi\|_1$$

$$\text{s.t.} \quad R_\Phi = T_{R \to \Phi} R$$

$$V(s) = R(s, a) + \gamma \sum_{s'} \theta_{sas'} V(s') \qquad \forall (s, a) \in \mathcal{D}$$

$$V(s) \geq R(s, a) + \gamma \sum_{s'} \theta_{sas'} V(s') + \varepsilon \qquad \forall s \in \mathcal{D}, (s, a) \notin \mathcal{D}$$

$$V(s) \geq R(s, a) + \gamma \sum_{s'} \theta_{sas'} V(s') \qquad \forall s \notin \mathcal{D}$$

This program can be solved efficiently with any quadratic programming solver. It contains on the order of $|\mathcal{S}||\mathcal{A}|$ variables and constraints, and the constraint matrix is sparse with $O(|\mathcal{S}||\mathcal{A}|\mu_a)$ non-zero entries, where $\mu_a$ is the average sparsity of $\theta_{sa}$ — that is, the average number of states $s'$ that have a non-zero probability of being reached from $s$ using action $a$. In our implementation, we use the `cvx` Matlab package [6] to solve this optimization efficiently.

## 4   Fitting Step

Once the reward function $R^{(i)}$ for the current feature set $\Phi^{(i-1)}$ is computed, we formulate a new feature hypothesis $\Phi^{(i)}$ that is better able to represent this reward function. The objective of this step is to construct a set of features that gives greater resolution in regions where the old features are too coarse, and lower resolution in regions where the old features are unnecessarily fine. We obtain $\Phi^{(i)}$ by building a regression tree for $R^{(i)}$ over the state-space $\mathcal{S}$, using the standard intra-cluster variance splitting criterion [3]. The tree is rooted at the node $t_0$, and each node of the tree is defined as $t_j = \{\delta_j, \phi_j, t_{j-}, t_{j+}\}$. $t_{j-}$ and $t_{j+}$ are the left and right subtrees, $\phi_j \subseteq \mathcal{S}$ is the set of states belonging to node $j$ (initialized as $\phi_0 = \mathcal{S}$), and $\delta_j$ is the component feature that acts as the splitting test at node $j$. States $s \in \phi_j$ for which $\delta_j(s) = 0$ are assigned to the left subtree, and states for which $\delta_j(s) = 1$ are assigned to the right subtree. In our implementation, all component features are binary, though the generalization to multivariate components and non-binary trees is straightforward. The new set of features consists of indicators for each of the leaf clusters $\phi_\ell$ (where $t_\ell$ is a leaf node), and can be equivalently expressed as a conjunction of components: letting $j_0, ..., j_n, \ell$ be the sequence of nodes on the path from the root to $t_\ell$, and defining $r_0, ..., r_n$ so that $r_k$ is 1 if $t_{j_{k+1}} = t_{j_k +}$ and 0 otherwise, $s \in \phi_\ell$ if and only if $\delta_{j_k}(s) = r_k$ for all $k \in \{0, ..., n\}$.

As discussed in Section 2, we prefer the smallest tree that produces a rich enough feature set to represent a reward function consistent with the examples $\mathcal{D}$. We therefore terminate the splitting procedure at node $t_\ell$ when we detect that further splitting of the node is unnecessary to maintain consistency with the example set. This is done by constructing a new reward function $\hat{R}^{(i)}$ for which $\hat{R}^{(i)}(s, a) = |\phi_\ell|^{-1} \sum_{s \in \phi_\ell} R^{(i)}(s, a)$ if $s \in \phi_\ell$, and $\hat{R}^{(i)}(s, a) = R^{(i)}(s, a)$ otherwise. The optimal policy under $\hat{R}^{(i)}$ is determined with value iteration and, if the policy is consistent with the examples $\mathcal{D}$, $t_\ell$ becomes a leaf and $R^{(i)}$ is updated to be equal to $\hat{R}^{(i)}$. Although value iteration ordinarily can take many iterations, since the changes we are considering often make small, local changes to the optimal policy compared to the current reward function $R^{(i)}$, we can often converge in only a few iterations by starting with the value function $V^{(i)}$ for the current reward $R^{(i)}$. We therefore store this value function and update it along with $R^{(i)}$.

In addition to this stopping criterion, we can also employ the loss factor $\Delta(\phi_{k_1}, \phi_{k_2})$ to encourage the next optimization step to assign similar values to nearby features, allowing them to be merged in subsequent iterations. Recall that $\Delta(\phi_{k_1}, \phi_{k_2})$ is a linear penalty on the difference between the average rewards of states in $\phi_{k_1}$ and $\phi_{k_2}$, and can be used to drive the rewards in these features closer together so that they can be merged in a subsequent iteration. Features found deeper in the tree exhibit greater complexity, since they are formed by a conjunction of a larger number of components. These complex features are more likely to be the result of overfitting, and can be merged to form smaller trees. To encourage such mergers, we set $\Delta(\phi_{k_1}, \phi_{k_2})$ to be proportional

| Gridworld size | Total states | LPAL (sec) | MMP (sec) | Abbeel & Ng (sec) | FIRL (sec total) | Optimization (sec each) | Fitting (sec each) |
|---|---|---|---|---|---|---|---|
| 16×16 | 256 | 0.29 | 0.24 | 27.05 | 8.34 | 0.39 | 0.11 |
| 32×32 | 1024 | 0.66 | 0.42 | 74.66 | 29.00 | 1.01 | 0.73 |
| 64×64 | 4096 | 2.22 | 1.26 | 272.10 | 165.29 | 4.26 | 5.80 |
| 128×128 | 16384 | 19.33 | 7.58 | 876.18 | 1208.47 | 24.44 | 48.44 |
| 256×256 | 65536 | 52.60 | 81.26 | 1339.87 | 10389.59 | 170.14 | 428.49 |

Table 1: Performance comparison of FIRL, LPAL, MMP, and Abbeel & Ng on gridworlds of varying size. FIRL ran for 15 iterations. Individual iterations were comparable in length to prior methods.

to the depth of the deepest common ancestor of $\phi_{k_1}$ and $\phi_{k_2}$. The loss factor is therefore set to $\Delta(\phi_{k_1}, \phi_{k_2}) = D_a(k_1, k_2)/D_t$, where $D_a$ gives the depth of the deepest common ancestor of two nodes, and $D_t$ is the total depth of the tree.

Finally, we found that limiting the depth of the tree and iteratively increasing that limit reduced overfitting and produced features that more accurately described the true reward function, since the optimization and fitting steps could communicate more frequently before committing to a set of complex features. We therefore begin with a depth limit of one, and increase the limit by one on each successive iteration. We experimented with a variety of other depth limiting schemes and found that this simple iterative deepening procedure produced the best results.

## 5 Experiments

### 5.1 Gridworld

In the first experiment, we compare FIRL with the MMP algorithm [9], the LPAL algorithm [10], and the algorithm of Abbeel & Ng [1] on a gridworld modeled after the one used by Abbeel & Ng. The purpose of this experiment is to determine how well FIRL performs on a standard IRL example, without knowledge of the relevant features. A gridworld consists of an $N{\times}N$ grid of states, with five actions possible in each state, corresponding to movement in each of the compass directions and standing in place. In the deterministic gridworld, each action deterministically moves the agent into the corresponding state. In the non-deterministic world, each action has a 30% chance of causing a transition to another random neighboring state. The world is partitioned into 64 equal-sized regions, and all the cells in a single region are assigned the same randomly selected reward. The expert's policy is the optimal policy under this reward. The example set $\mathcal{D}$ is generated by randomly sampling states and following the expert's policy for 100 steps.

Since the prior algorithms do not perform feature construction, they were tested either with indicators for each of the 64 regions (referred to as "perfect" features), or with indicators for each state (the "primitive" features). FIRL was instead provided with $2N$ component features corresponding to splits on the $x$ and $y$ axes, so that $\delta_{x,i}(s_{x,y}) = 1$ if $x \geq i$, and $\delta_{y,i}(s_{x,y}) = 1$ if $y \geq i$. By composing such splits, it is possible to represent any rectangular partitioning of the state space.

We first compare the running times of the algorithms (using perfect features for prior methods) on gridworlds of varying sizes, shown in Table 1. Performance was tested on an Intel Core i7 2.66 GHz computer. Each trial was repeated 10 times on random gridworlds, with average running times presented. For FIRL, running time is given for 15 iterations, and is also broken down into the average length of each optimization and fitting step. Although FIRL is often slower than methods that do not perform feature construction, the results suggest that it scales gracefully with the size of the problem. The optimization time scales almost linearly, while the tree construction scales worse than linearly but better than quadratically. The latter can likely be improved for large problems by using heuristics to minimize evaluations of the expensive stopping test.

In the second experiment, shown in Figure 1, we evaluate accuracy on $64 \times 64$ gridworlds with varying numbers of examples, again repeating each trial 10 times. We measured the percentage of states in which each algorithm failed to predict the expert's optimal action ("percent misprediction"), as well as the Euclidean distance between the expectations of the perfect features under the learned policy and the expert's policy (normalized by $(1 - \gamma)$ as suggested by Abbeel & Ng [1]). For the mixed policies produced by Abbeel & Ng, we computed the metrics for each policy and mixed them using the policy weights $\lambda$ [1]. For the non-deterministic policies of LPAL, percent misprediction is

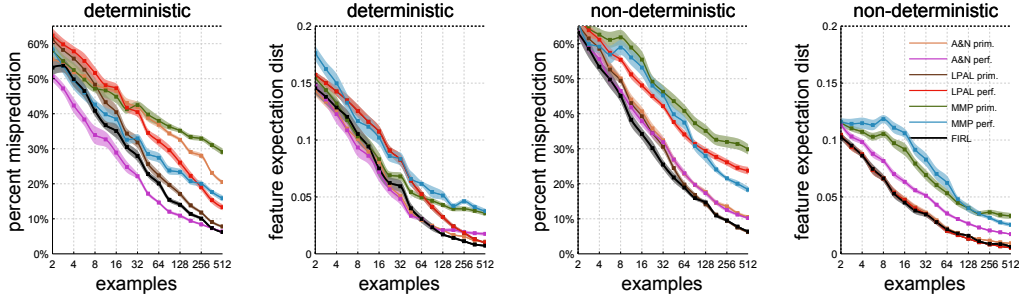

Figure 1: Accuracy comparison between FIRL, LPAL, MMP, and Abbeel & Ng, the latter provided with either perfect or primitive features. Shaded regions show standard error. Although FIRL was not provided the perfect features, it achieved similar accuracy to prior methods that were.

the mean probability of taking an incorrect action in each state. Results for prior methods are shown with both the perfect and primitive features. FIRL again ran for 15 iterations, and generally achieved comparable accuracy to prior algorithms, even when they were provided with perfect features.

## 5.2 Transfer Between Environments

While the gridworld experiments demonstrate that FIRL performs comparably to existing methods on this standard example, even without knowing the correct features, they do not evaluate the two key advantages of FIRL: its ability to construct features from primitive components, and its ability to generalize learned rewards to different environments. To evaluate reward transfer and see how the method performs with more realistic component features, we populated a world with objects. This environment also consists of an $N \times N$ grid of states, with the same actions as the gridworld. Objects are randomly placed with 5% probability in each state, and each object has 1 of $C$ "inner" and "outer" colors, selected uniformly at random. The algorithm was provided with components of the form "is the nearest $X$ at most $n$ units away," where $X$ is a wall or an object with a specific inner or outer color, giving a total of $(2C + 1)N$ component features. The expert received a reward of $-2$ for being within 3 units of an object with inner color 1, otherwise a reward of $-1$ for being within 2 units of a wall, otherwise a reward of 1 for being within 1 unit of an object with inner color 2, and 0 otherwise. All other colors acted as distractors, allowing us to evaluate the robustness of feature construction to irrelevant components. For each trial, the learned reward tree was used to test accuracy on 10 more random environments, by specifying a reward for each state according to the regression tree. We will refer to these experiments as "transfer." Each trial was repeated 10 times.

In Figure 2(a), we evaluate how FIRL performs with varying numbers of iterations on both the training and transfer environments, as well as on the gridworld from the previous section. The results indicate that FIRL converged to a stable hypothesis more quickly than in the gridworld, since the square regions in the gridworld required many more partitions than the object-relative features. However, the required number of iterations was low on both environments.

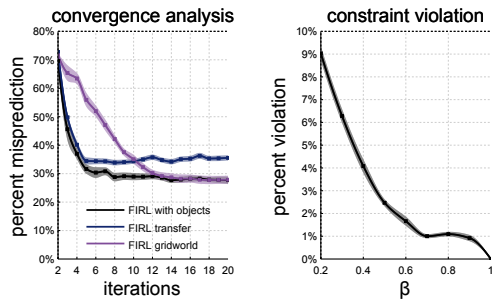

Figure 2(a): FIRL converged after a small number of iterations.

Figure 2(b): Constraint violation was low in non-deterministic MDPs.

In Figure 2(b), we evaluate how often the non-convex constraints discussed in Section 3 are violated under our convex approximation. We measure the percent of examples that are violated with varying amounts of non-determinism, by varying the probability $\beta$ with which an action moves the agent to the desired state. $\beta = 1$ is deterministic, and $\beta = 0.2$ gives a uniform distribution over neighboring states. The results suggest that the constraint is rarely violated under the convex relaxation, even in highly non-deterministic MDPs, and the number of violations decreases sharply as the MDP becomes more deterministic.

We compared FIRL's accuracy on the transfer task with Abbeel & Ng and MMP. LPAL was not used in the comparison because it does not return a reward function, and therefore cannot transfer

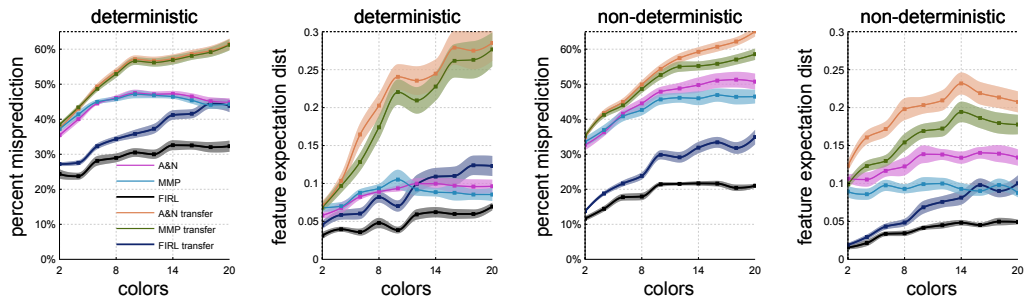

Figure 3: Comparison of FIRL and Abbeel & Ng on training environments and randomly generated transfer environments, with increasing numbers of component features. FIRL maintained higher transfer accuracy in the presence of distractors by constructing features out of relevant components.

its policy to new environments. Since prior methods do not perform feature construction, they were provided with all of the component features. The experiments used $64 \times 64$ environments and 64 examples. The number of colors $C$ was varied from 2 to 20 to test how well the algorithms handle irrelevant "distractors." FIRL ran for 10 iterations on each trial. The results in Figure 3 indicate that accuracy on the training environment remained largely stable, while transfer accuracy gradually decreased with more colors due to the ambiguity caused by large numbers of distractors. Prior algorithms were more affected by distractors on the training environments, and their inability to construct features prevented them from capturing a portable "explanation" of the expert's reward. They therefore could not transfer the learned policy to other environments with comparable accuracy. In contrast to the gridworld experiments, the expert's reward function in these environments was encoded in terms of logical relationships between the component features, which standard IRL algorithms cannot capture. In the next section, we examine another environment that also exemplifies the need for feature construction.

## 5.3 Highway Driving Behaviors

To demonstrate FIRL's ability to learn meaningful behaviors, we implemented a driving simulator inspired by the environments in [1] and [10]. The task is to navigate a car on a three-lane highway. All other vehicles are moving at speed 1. The agent can drive at speeds 1 through 4, and can move one lane left or one lane right. The other vehicles can be cars or motorcycles, and can be either civilian or police, for a total of 4 possibilities. The component features take the form "is a vehicle of type $X$ at most $n$ car-lengths in front/behind me," where $X$ can be either all vehicles, cars, motorcycles, police, or civilian, and $n$ is in the range from 0 to 5 car-lengths. There are equivalent features for checking for cars in front or behind in the lanes to the left and to the right of the agent's, as well as a feature for each of the four speeds and each lane the agent can occupy.

The rich feature set of this driving simulator enables interesting behaviors to be demonstrated. For this experiment, we implemented expert policies for two behaviors: a "lawful" driver and an "outlaw" driver. The lawful driver prefers to drive fast, but does not exceed speed 2 in the right lane, or speed 3 in the middle lane. The outlaw driver also prefers to drive fast, but slows down to speed 2 or below when within 2 car-lengths of a police vehicle (to avoid arrest).

In Table 2, we compare the policies learned from traces of the two experts by FIRL, MMP, and Abbeel & Ng's algorithm. As before, prior methods were provided with all of the component features. All algorithms were trained on 30 traces on a stretch of highway 100 car-lengths long, and tested on 10 novel highways. As can be seen in the supplemental videos, the policy learned by FIRL closely matched that of the expert, maintaining a high speed whenever possible but not driving fast in the wrong lane or near police vehicles. The policies learned by Abbeel & Ng's algorithm and MMP drove at the minimum speed when trained on either the lawful or outlaw expert traces. Because prior methods only represented the reward as a linear combination of the provided features, they were unable to determine the logical connection between speed and the other features. The policies learned by these methods found the nearest "optimal" position with respect to their learned feature weights, accepting the cost of violating the speed expectation in exchange for best matching the expectation of all other (largely irrelevant) features. FIRL, on the other hand, correctly established

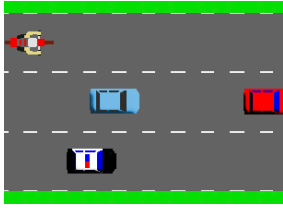

| | "Lawful" policies | | | "Outlaw" policies | | |
|---|---|---|---|---|---|---|
| | percent mis-prediction | feature expect-ation distance | average speed | percent mis-prediction | feature expect-ation distance | average speed |
| Expert | 0.0% | 0.000 | 2.410 | 0.0% | 0.000 | 2.375 |
| FIRL | 22.9% | 0.025 | 2.314 | 24.2% | 0.027 | 2.376 |
| MMP | 27.0% | 0.111 | 1.068 | 27.2% | 0.096 | 1.056 |
| A&N | 38.6% | 0.202 | 1.054 | 39.3% | 0.164 | 1.055 |
| Random | 42.7% | 0.220 | 1.053 | 41.4% | 0.184 | 1.053 |

Table 2: Comparison of FIRL, MMP and Abbeel & Ng on the highway environment (left). The policies learned by FIRL closely match the expert's average speed, while those of other methods do not. The difference between the policies is particularly apparent in the supplemental videos, which can be found at `http://graphics.stanford.edu/projects/firl/index.htm`

the logical connection between speed and police vehicles or lanes, and drove fast when appropriate, as indicated by the average speed in Table 2. As a baseline, the table also shows the performance of a random policy generated by picking weights for the component features uniformly at random.

## 6   Discussion and Future Work

This paper presents an IRL algorithm that constructs reward features, represented as a regression tree, out of a large collection of component features. By combining relevant components into logical conjunctions, the FIRL algorithm is able to discover logical precedence relationships that would not otherwise be apparent. The learned regression tree concisely captures the structure of the reward function and acts as a portable "explanation" of the observed behavior in terms of the provided components, allowing the learned reward function to be transplanted onto different environments.

Feature construction for IRL may be a valuable tool for analyzing the motivations of an agent (such as a human or an animal) from observed behavior. Research indicates that animals learn optimal policies for a pattern of rewards [4], suggesting that it may be possible to learn such behavior with IRL. While it can be difficult to manually construct a complete list of relevant reward features for such an agent, it is comparatively easier to list all aspects of the environment that a human or animal is aware of. With FIRL, such a list can be used to form hypotheses about reward features, possibly leading to increased understanding of the agent's motivations. In fact, models that perform a variant of IRL have been shown to correspond well to goal inference in humans [2].

While FIRL achieves good performance on discrete MDPs, in its present form it is unable to handle continuous state spaces, since the optimization constraints require an enumeration of all states in $\mathcal{S}$. Approximate linear programming has been used to solve MDPs with continuous state spaces [5], and a similar approach could be used to construct a tractable set of constraints for the optimization step, making it possible to perform feature construction on continuous or extremely large state spaces.

Although we found that FIRL converged to a stable hypothesis quickly, it is difficult to provide an accurate convergence test. Theoretical analysis of convergence is complicated by the fact that regression trees provide few guarantees. The conventional training error metric is not a good measure of convergence, because the optimization constraints keep training error consistently low. Instead, we can use cross-validation, or heuristics such as leaf count and tree depth, to estimate convergence. In practice, we found this unnecessary, as FIRL consistently converged in very few iterations. Defining a practical convergence test and analyzing convergence is an interesting avenue for future work.

FIRL may also benefit from future work on the fitting step. A more intelligent hypothesis proposal scheme, perhaps with a Bayesian approach, could more readily incorporate priors on potential features to penalize excessively deep trees or prevent improbable conjunctions of components. Furthermore, while regression trees provide a principled method for constructing logical conjunctions of component features, if the desired features are not readily expressible as conjunctions of simple components, other regression methods may be used in the fitting step. For example, the algorithm could be modified to perform feature adaptation by using the fitting step to adapt a set of continuously-parameterized features to best fit the reward function.

**Acknowledgments.**   We thank Andrew Y. Ng, Emanuel Todorov, and Sameer Agarwal for helpful feedback and discussion. This work was supported in part by NSF grant CCF-0641402.

**References**

[1] P. Abbeel and A. Y. Ng. Apprenticeship learning via inverse reinforcement learning. In *ICML '04: Proceedings of the 21st International Conference on Machine Learning*. ACM, 2004.

[2] C. L. Baker, J. B. Tenenbaum, and R. R. Saxe. Goal inference as inverse planning. In *Proceedings of the 29th Annual Conference of the Cognitive Science Society*, 2007.

[3] L. Breiman, J. Friedman, R. Olshen, and C. Stone. *Classification and Regression Trees*. Wadsworth and Brooks, Monterey, CA, 1984.

[4] P. Dayan and B. W. Balleine. Reward, motivation, and reinforcement learning. *Neuron*, 36(2):285–298, 2002.

[5] D. P. de Farias and B. Van Roy. The linear programming approach to approximate dynamic programming. *Operations Research*, 51(6):850–865, 2003.

[6] M. Grant and S. Boyd. CVX: Matlab Software for Disciplined Convex Programming (web page and software), 2008. `http://stanford.edu/~boyd/cvx`.

[7] A. Y. Ng and S. J. Russell. Algorithms for inverse reinforcement learning. In *ICML '00: Proceedings of the 17th International Conference on Machine Learning*, pages 663–670. Morgan Kaufmann Publishers Inc., 2000.

[8] D. Ramachandran and E. Amir. Bayesian inverse reinforcement learning. In *IJCAI'07: Proceedings of the 20th International Joint Conference on Artifical Intelligence*, pages 2586–2591. Morgan Kaufmann Publishers Inc., 2007.

[9] N. D. Ratliff, J. A. Bagnell, and M. A. Zinkevich. Maximum margin planning. In *ICML '06: Proceedings of the 23rd International Conference on Machine Learning*, pages 729–736. ACM, 2006.

[10] U. Syed, M. Bowling, and R. E. Schapire. Apprenticeship learning using linear programming. In *ICML '08: Proceedings of the 25th International Conference on Machine Learning*, pages 1032–1039. ACM, 2008.

